# Estimating Average-Case Learning Curves Using Bayesian, Statistical Physics and VC Dimension Methods

**David Haussler**
University of California
Santa Cruz, California

**Michael Kearns***
AT&T Bell Laboratories
Murray Hill, New Jersey

**Manfred Opper**
Institut für Theoretische Physik
Universität Giessen, Germany

**Robert Schapire**
AT&T Bell Laboratories
Murray Hill, New Jersey

## Abstract

In this paper we investigate an average-case model of concept learning, and give results that place the popular statistical physics and VC dimension theories of learning curve behavior in a common framework.

## 1 INTRODUCTION

In this paper we study a simple concept learning model in which the learner attempts to infer an unknown *target concept* $f$, chosen from a known *concept class* $\mathcal{F}$ of $\{0, 1\}$-valued functions over an input space $X$. At each trial $i$, the learner is given a point $x_i \in X$ and asked to predict the value of $f(x_i)$. If the learner predicts $f(x_i)$ incorrectly, we say the learner makes a *mistake*. After making its prediction, the learner is told the correct value.

This simple theoretical paradigm applies to many areas of machine learning, including much of the research in neural networks. The quantity of fundamental interest in this setting is the *learning curve*, which is the function of $m$ defined as the prob-

ability the learning algorithm makes a mistake predicting $f(x_{m+1})$, having already seen the examples $(x_1, f(x_1)), \ldots, (x_m, f(x_m))$.

In this paper we study learning curves in an average-case setting that admits a prior distribution over the concepts in $\mathcal{F}$. We examine learning curve behavior for the optimal *Bayes* algorithm and for the related *Gibbs* algorithm that has been studied in statistical physics analyses of learning curve behavior. For both algorithms we give new upper and lower bounds on the learning curve in terms of the Shannon information gain.

The main contribution of this research is in showing that the average-case or Bayesian model provides a unifying framework for the popular statistical physics and VC dimension theories of learning curves. By beginning in an average-case setting and deriving bounds in information-theoretic terms, we can gradually recover a worst-case theory by removing the averaging in favor of combinatorial parameters that upper bound certain expectations.

Due to space limitations, the paper is technically dense and almost all derivations and proofs have been omitted. We strongly encourage the reader to refer to our longer and more complete versions [4, 6] for additional motivation and technical detail.

## 2    NOTATIONAL CONVENTIONS

Let $X$ be a set called the *instance space*. A *concept class* $\mathcal{F}$ over $X$ is a (possibly infinite) collection of subsets of $X$. We will find it convenient to view a concept $f \in \mathcal{F}$ as a function $f : X \to \{0, 1\}$, where we interpret $f(x) = 1$ to mean that $x \in X$ is a *positive example* of $f$, and $f(x) = 0$ to mean $x$ is a *negative example*.

The symbols $\mathcal{P}$ and $\mathcal{D}$ are used to denote probability distributions. The distribution $\mathcal{P}$ is over $\mathcal{F}$, and $\mathcal{D}$ is over $X$. When $\mathcal{F}$ and $X$ are countable we assume that these distributions are defined as probability mass functions. For uncountable $\mathcal{F}$ and $X$ they are assumed to be probability measures over some appropriate $\sigma$-algebra. All of our results hold for both countable and uncountable $\mathcal{F}$ and $X$.

We use the notation $\mathbf{E}_{f \in \mathcal{P}}[\chi(f)]$ for the expectation of the random variable $\chi$ with respect to the distribution $\mathcal{P}$, and $\mathbf{Pr}_{f \in \mathcal{P}}[cond(f)]$ for the probability with respect to the distribution $\mathcal{P}$ of the set of all $f$ satisfying the predicate $cond(f)$. Everything that needs to be measurable is assumed to be measurable.

## 3    INFORMATION GAIN AND LEARNING

Let $\mathcal{F}$ be a concept class over the instance space $X$. Fix a *target concept* $f \in \mathcal{F}$ and an infinite sequence of instances $\mathbf{x} = x_1, \ldots, x_m, x_{m+1}, \ldots$ with $x_m \in X$ for all $m$. For now we assume that the fixed instance sequence $\mathbf{x}$ is known in advance to the learner, but that the target concept $f$ is not. Let $\mathcal{P}$ be a probability distribution over the concept class $\mathcal{F}$. We think of $\mathcal{P}$ in the Bayesian sense as representing the *prior beliefs* of the learner about which target concept it will be learning.

In our setting, the learner receives information about $f$ incrementally via the label

sequence $f(x_1), \ldots, f(x_m), f(x_{m+1}), \ldots$. At time $m$, the learner receives the label $f(x_m)$. For any $m \geq 1$ we define (with respect to $\mathbf{x}, f$) the $m$th *version space*

$$\mathcal{F}_m(\mathbf{x}, f) = \{\hat{f} \in \mathcal{F} : \hat{f}(x_1) = f(x_1), \ldots, \hat{f}(x_m) = f(x_m)\}$$

and the $m$th *volume* $V_m^{\mathcal{P}}(\mathbf{x}, f) = \mathcal{P}[\mathcal{F}_m(\mathbf{x}, f)]$. We define $\mathcal{F}_0(\mathbf{x}, f) = \mathcal{F}$ for all $\mathbf{x}$ and $f$, so $V_0^{\mathcal{P}}(\mathbf{x}, f) = 1$. The version space at time $m$ is simply the class of all concepts in $\mathcal{F}$ consistent with the first $m$ labels of $f$ (with respect to $\mathbf{x}$), and the $m$th volume is the measure of this class under $\mathcal{P}$. For the first part of the paper, the infinite instance sequence $\mathbf{x}$ and the prior $\mathcal{P}$ are fixed; thus we simply write $\mathcal{F}_m(f)$ and $V_m(f)$. Later, when the sequence $\mathbf{x}$ is chosen randomly, we will reintroduce this dependence explicitly. We adopt this notational practice of omitting any dependence on a fixed $\mathbf{x}$ in many other places as well.

For each $m \geq 0$ let us define the $m$th *posterior distribution* $\mathcal{P}_m(\mathbf{x}, f) = \mathcal{P}_m$ by restricting $\mathcal{P}$ to the $m$th version space $\mathcal{F}_m(f)$; that is, for all (measurable) $S \subset \mathcal{F}$, $\mathcal{P}_m[S] = \mathcal{P}[S \cap \mathcal{F}_m(f)]/\mathcal{P}[\mathcal{F}_m(f)] = \mathcal{P}[S \cap \mathcal{F}_m(f)]/V_m(f)$.

Having already seen $f(x_1), \ldots, f(x_m)$, how much information (assuming the prior $\mathcal{P}$) does the learner expect to gain by seeing $f(x_{m+1})$? If we let $\mathcal{I}_{m+1}(\mathbf{x}, f)$ (abbreviated $\mathcal{I}_{m+1}(f)$ since $\mathbf{x}$ is fixed for now) be a random variable whose value is the (Shannon) information gained from $f(x_{m+1})$, then it can be shown that the expected information is

$$\mathbf{E}_{f \in \mathcal{P}}[\mathcal{I}_{m+1}(f)] = \mathbf{E}_{f \in \mathcal{P}}\left[-\log \frac{V_{m+1}(f)}{V_m(f)}\right] = \mathbf{E}_{f \in \mathcal{P}}[-\log \chi_{m+1}(f)] \qquad (1)$$

where we define the $(m + 1)$st *volume ratio* by $\chi_{m+1}^{\mathcal{P}}(\mathbf{x}, f) = \chi_{m+1}(f) = V_{m+1}(f)/V_m(f)$.

We now return to our learning problem, which we define to be that of predicting the label $f(x_{m+1})$ given only the previous labels $f(x_1), \ldots, f(x_m)$. The first learning algorithm we consider is called the *Bayes optimal classification algorithm*, or the *Bayes* algorithm for short. For any $m$ and $b \in \{0, 1\}$, define $\mathcal{F}_m^b(\mathbf{x}, f) = \mathcal{F}_m^b(f) = \{\hat{f} \in \mathcal{F}_m(\mathbf{x}, f) : \hat{f}(x_{m+1}) = b\}$. Then the Bayes algorithm is:

If $\mathcal{P}_m[\mathcal{F}_m^1(f)] > \mathcal{P}_m[\mathcal{F}_m^0(f)]$, predict $f(x_{m+1}) = 1$.

If $\mathcal{P}_m[\mathcal{F}_m^1(f)] < \mathcal{P}_m[\mathcal{F}_m^0(f)]$, predict $f(x_{m+1}) = 0$.

If $\mathcal{P}_m[\mathcal{F}_m^1(f)] = \mathcal{P}_m[\mathcal{F}_m^0(f)]$, flip a fair coin to predict $f(x_{m+1})$.

It is well known that if the target concept $f$ is drawn at random according to the prior distribution $\mathcal{P}$, then the Bayes algorithm is optimal in the sense that it minimizes the probability that $f(x_{m+1})$ is predicted incorrectly. Furthermore, if we let $Bayes_{m+1}^{\mathcal{P}}(\mathbf{x}, f)$ (abbreviated $Bayes_{m+1}^{\mathcal{P}}(f)$ since $\mathbf{x}$ is fixed for now) be a random variable whose value is 1 if the Bayes algorithm predicts $f(x_{m+1})$ correctly and 0 otherwise, then it can be shown that the probability of a mistake for a random $f$ is

$$\mathbf{E}_{f \in \mathcal{P}}[Bayes_{m+1}^{\mathcal{P}}(f)] = \mathbf{E}_{f \in \mathcal{P}}\left[\Theta\left(\frac{1}{2} - \chi_{m+1}(f)\right)\right]. \qquad (2)$$

Despite the optimality of the Bayes algorithm, it suffers the drawback that its *hypothesis* at any time $m$ may not be a member of the target class $\mathcal{F}$. (Here we

define the hypothesis of an algorithm at time $m$ to be the (possibly probabilistic) mapping $\hat{f} : X \rightarrow \{0, 1\}$ obtained by letting $\hat{f}(x)$ be the prediction of the algorithm when $x_{m+1} = x$.) This drawback is absent in our second learning algorithm, which we call the *Gibbs* algorithm [6]:

Given $f(x_1), \ldots, f(x_m)$, choose a hypothesis concept $\hat{f}$ randomly from $\mathcal{P}_m$.

Given $x_{m+1}$, predict $f(x_{m+1}) = \hat{f}(x_{m+1})$.

The Gibbs algorithm is the "zero-temperature" limit of the learning algorithm studied in several recent papers [2, 3, 8, 9]. If we let $Gibbs_{m+1}^{\mathcal{P}}(\mathbf{x}, f)$ (abbreviated $Gibbs_{m+1}^{\mathcal{P}}(f)$ since $\mathbf{x}$ is fixed for now) be a random variable whose value is 1 if the Gibbs algorithm predicts $f(x_{m+1})$ correctly and 0 otherwise, then it can be shown that the probability of a mistake for a random $f$ is

$$\mathbf{E}_{f \in \mathcal{P}}[Gibbs_{m+1}^{\mathcal{P}}(f)] = \mathbf{E}_{f \in \mathcal{P}}[1 - \chi_{m+1}(f)]. \tag{3}$$

Note that by the definition of the Gibbs algorithm, Equation (3) is exactly the average probability of mistake of a consistent hypothesis, using the distribution on $\mathcal{F}$ defined by the prior. Thus bounds on this expectation provide an interesting contrast to those obtained via VC dimension analysis, which always gives bounds on the probability of mistake of the *worst* consistent hypothesis.

## 4    THE MAIN INEQUALITY

In this section we state one of our main results: a chain of inequalities that upper and lower bounds the expected error for both the Bayes and Gibbs algorithms by simple functions of the expected information gain. More precisely, using the characterizations of the expectations in terms of the volume ratio $\chi_{m+1}(f)$ given by Equations (1), (2) and (3), we can prove the following, which we refer to as the main inequality:

$$\begin{aligned} \mathcal{H}^{-1}(\mathbf{E}_{f \in \mathcal{P}}[\mathcal{I}_{m+1}(f)]) &\leq \mathbf{E}_{f \in \mathcal{P}}[Bayes_{m+1}(f)] \\ &\leq \mathbf{E}_{f \in \mathcal{P}}[Gibbs_{m+1}(f)] \leq \frac{1}{2}\mathbf{E}_{f \in \mathcal{P}}[\mathcal{I}_{m+1}(f)]. \end{aligned} \tag{4}$$

Here we have defined an inverse to the binary entropy function $\mathcal{H}(p) = -p \log p - (1 - p) \log(1 - p)$ by letting $\mathcal{H}^{-1}(q)$, for $q \in [0, 1]$, be the unique $p \in [0, 1/2]$ such that $\mathcal{H}(p) = q$. Note that the bounds given depend on properties of the particular prior $\mathcal{P}$, and on properties of the particular fixed sequence $\mathbf{x}$. These upper and lower bounds are equal (and therefore tight) at both extremes $\mathbf{E}_{f \in \mathcal{P}}[\mathcal{I}_{m+1}(f)] = 1$ (maximal information gain) and $\mathbf{E}_{f \in \mathcal{P}}[\mathcal{I}_{m+1}(f)] = 0$ (minimal information gain). To obtain a weaker but perhaps more convenient lower bound, it can also be shown that there is a constant $c_0 > 0$ such that for all $p > 0$, $\mathcal{H}^{-1}(p) \geq c_0 p / \log(2/p)$.

Finally, if all that is wanted is a direct comparison of the performances of the Gibbs and Bayes algorithms, we can also show:

$$\mathbf{E}_{f \in \mathcal{P}}[Bayes_{m+1}(f)] \leq \mathbf{E}_{f \in \mathcal{P}}[Gibbs_{m+1}(f)] \leq 2\mathbf{E}_{f \in \mathcal{P}}[Bayes_{m+1}(f)]. \tag{5}$$

## 5   THE MAIN INEQUALITY: CUMULATIVE VERSION

In this section we state a cumulative version of the main inequality: namely, bounds on the expected *cumulative* number of mistakes made in the first $m$ trials (rather than just the instantaneous expectations).

First, for the cumulative information gain, it can be shown that $\mathbf{E}_{f \in \mathcal{P}}[\sum_{i=1}^{m} \mathcal{I}_i(f)] = \mathbf{E}_{f \in \mathcal{P}}[-\log V_m(f)]$. This expression has a natural interpretation. The first $m$ instances $x_1, \ldots, x_m$ of $\mathbf{x}$ induce a partition $\Pi_m^{\mathcal{F}}(\mathbf{x})$ of the concept class $\mathcal{F}$ defined by $\Pi_m^{\mathcal{F}}(\mathbf{x}) = \Pi_m^{\mathcal{F}} = \{\mathcal{F}_m(\mathbf{x}, f) : f \in \mathcal{F}\}$. Note that $|\Pi_m^{\mathcal{F}}|$ is always at most $2^m$, but may be considerably smaller, depending on the interaction between $\mathcal{F}$ and $x_1, \ldots, x_m$. It is clear that $\mathbf{E}_{f \in \mathcal{P}}[-\log V_m(f)] = -\sum_{\pi \in \Pi_m^{\mathcal{F}}} \mathcal{P}[\pi] \log \mathcal{P}[\pi]$. Thus the expected cumulative information gained from the labels of $x_1, \ldots, x_m$ is simply the entropy of the partition $\Pi_m^{\mathcal{F}}$ under the distribution $\mathcal{P}$. We shall denote this entropy by $\mathcal{H}^{\mathcal{P}}(\Pi_m^{\mathcal{F}}(\mathbf{x})) = \mathcal{H}_m^{\mathcal{P}}(\mathbf{x}) = \mathcal{H}_m^{\mathcal{P}}$. Now analogous to the main inequality for the instantaneous case (Inequality (4)), we can show:

$$\frac{c_0 \mathcal{H}_m^{\mathcal{P}}}{\log(2m/\mathcal{H}_m^{\mathcal{P}})} \leq m\mathcal{H}^{-1}\left(\frac{1}{m}\mathcal{H}_m^{\mathcal{P}}\right) \leq \mathbf{E}_{f \in \mathcal{P}}\left[\sum_{i=1}^{m} Bayes_i(f)\right]$$

$$\leq \mathbf{E}_{f \in \mathcal{P}}\left[\sum_{i=1}^{m} Gibbs_i(f)\right] \leq \frac{1}{2}\mathcal{H}_m^{\mathcal{P}}. \tag{6}$$

Here we have applied the inequality $\mathcal{H}^{-1}(p) \geq c_0 p/\log(2/p)$ in order to give the lower bound in more convenient form. As in the instantaneous case, the upper and lower bounds here depend on properties of the particular $\mathcal{P}$ and $\mathbf{x}$. When the cumulative information gain is maximum ($\mathcal{H}_m^{\mathcal{P}} = m$), the upper and lower bounds are tight.

These bounds on learning performance in terms of a partition entropy are of special importance to us, since they will form the crucial link between the Bayesian setting and the Vapnik-Chervonenkis dimension theory.

## 6   MOVING TO A WORST-CASE THEORY: BOUNDING THE INFORMATION GAIN BY THE VC DIMENSION

Although we have given upper bounds on the expected cumulative number of mistakes for the Bayes and Gibbs algorithms in terms of $\mathcal{H}_m^{\mathcal{P}}(\mathbf{x})$, we are still left with the problem of evaluating this entropy, or at least obtaining reasonable upper bounds on it. We can intuitively see that the "worst case" for learning occurs when the partition entropy $\mathcal{H}_m^{\mathcal{P}}(\mathbf{x})$ is as large as possible. In our context, the entropy is qualitatively maximized when two conditions hold: (1) the instance sequence $\mathbf{x}$ induces a partition of $\mathcal{F}$ that is the largest possible, and (2) the prior $\mathcal{P}$ gives equal weight to each element of this partition.

In this section, we move away from our Bayesian average-case setting to obtain worst-case bounds by formalizing these two conditions in terms of combinatorial parameters depending only on the concept class $\mathcal{F}$. In doing so, we form the link between the theory developed so far and the VC dimension theory.

The second of the two conditions above is easily quantified. Since the entropy of a partition is at most the logarithm of the number of classes in it, a trivial upper bound on the entropy which holds for all priors $\mathcal{P}$ is $\mathcal{H}_m^{\mathcal{P}}(\mathbf{x}) \leq \log |\Pi_m^{\mathcal{F}}(\mathbf{x})|$. VC dimension theory provides an upper bound on $\log |\Pi_m^{\mathcal{F}}(\mathbf{x})|$ as follows.

For any sequence $\mathbf{x} = x_1, x_2, \ldots$ of instances and for $m \geq 1$, let $\dim_m(\mathcal{F}, \mathbf{x})$ denote the largest $d \geq 0$ such that there exists a subsequence $x_{i_1}, \ldots, x_{i_d}$ of $x_1, \ldots, x_m$ with $|\Pi_m^{\mathcal{F}}(\langle x_{i_1}, \ldots, x_{i_d} \rangle)| = 2^d$; that is, for every possible labeling of $x_{i_1}, \ldots, x_{i_d}$ there is some target concept in $\mathcal{F}$ that gives this labeling. The *Vapnik-Chervonenkis (VC) dimension* of $\mathcal{F}$ is defined by $\dim(\mathcal{F}) = \max\{\dim_m(\mathcal{F}, \mathbf{x}) : m \geq 1 \text{ and } x_1, x_2, \ldots \in X\}$. It can be shown [7, 10] that for all $\mathbf{x}$ and $m \geq d \geq 1$,

$$\log |\Pi_m^{\mathcal{F}}(\mathbf{x})| \leq (1 + o(1)) \dim_m(\mathcal{F}, \mathbf{x}) \log \frac{m}{\dim_m(\mathcal{F}, \mathbf{x})} \tag{7}$$

where $o(1)$ is a quantity that goes to zero as $\alpha = m/\dim_m(\mathcal{F}, \mathbf{x})$ goes to infinity.

In all of our discussions so far, we have assumed that the instance sequence $\mathbf{x}$ is fixed in advance, but that the target concept $f$ is drawn randomly according to $\mathcal{P}$. We now move to the completely probabilistic model, in which $f$ is drawn according to $\mathcal{P}$, and each instance $x_m$ in the sequence $\mathbf{x}$ is drawn randomly and independently according to a distribution $\mathcal{D}$ over the instance space $X$ (this infinite sequence of draws from $\mathcal{D}$ will be denoted $\mathbf{x} \in \mathcal{D}^*$). Under these assumptions, it follows from Inequalities (6) and (7), and the observation above that $\mathcal{H}_m^{\mathcal{P}}(\mathbf{x}) \leq \log |\Pi_m^{\mathcal{F}}(\mathbf{x})|$ that for any $\mathcal{P}$ and any $\mathcal{D}$,

$$
\begin{aligned}
\mathbf{E}_{f \in \mathcal{P}, \mathbf{x} \in \mathcal{D}^*} \left[ \sum_{i=1}^{m} Bayes_i(\mathbf{x}, f) \right] &\leq \mathbf{E}_{f \in \mathcal{P}, \mathbf{x} \in \mathcal{D}^*} \left[ \sum_{i=1}^{m} Gibbs_i(\mathbf{x}, f) \right] \\
&\leq \frac{1}{2} \mathbf{E}_{\mathbf{x} \in \mathcal{D}^*} [\log |\Pi_m^{\mathcal{F}}(\mathbf{x})|] \\
&\leq (1 + o(1)) \mathbf{E}_{\mathbf{x} \in \mathcal{D}^*} \left[ \frac{\dim_m(\mathcal{F}, \mathbf{x})}{2} \log \frac{m}{\dim_m(\mathcal{F}, \mathbf{x})} \right] \\
&\leq (1 + o(1)) \frac{\dim(\mathcal{F})}{2} \log \frac{m}{\dim(\mathcal{F})}. \tag{8}
\end{aligned}
$$

The expectation $\mathbf{E}_{\mathbf{x} \in \mathcal{D}^*} [\log |\Pi_m^{\mathcal{F}}(\mathbf{x})|]$ is the *VC entropy* defined by Vapnik and Chervonenkis in their seminal paper on uniform convergence [11].

In terms of instantaneous mistake bounds, using more sophisticated techniques [4], we can show that for any $\mathcal{P}$ and any $\mathcal{D}$,

$$\mathbf{E}_{f \in \mathcal{P}, \mathbf{x} \in \mathcal{D}^*}[Bayes_m(\mathbf{x}, f)] \leq \mathbf{E}_{\mathbf{x} \in \mathcal{D}^*} \left[ \frac{\dim_m(\mathcal{F}, \mathbf{x})}{m} \right] \leq \frac{\dim(\mathcal{F})}{m} \tag{9}$$

$$\mathbf{E}_{f \in \mathcal{P}, \mathbf{x} \in \mathcal{D}^*}[Gibbs_m(\mathbf{x}, f)] \leq \mathbf{E}_{\mathbf{x} \in \mathcal{D}^*} \left[ \frac{2\dim_m(\mathcal{F}, \mathbf{x})}{m} \right] \leq \frac{2\dim(\mathcal{F})}{m} \tag{10}$$

Haussler, Littlestone and Warmuth [5] construct specific $\mathcal{D}$, $\mathcal{P}$ and $\mathcal{F}$ for which the last bound given by Inequality (8) is tight to within a factor of $1/\ln(2) \approx 1.44$; thus this bound cannot be improved by more than this factor in general.[1] Similarly, the

bound given by Inequality (9) cannot be improved by more than a factor of 2 in general.

For specific $\mathcal{D}$, $\mathcal{P}$ and $\mathcal{F}$, however, it is possible to improve the general bounds given in Inequalities (8), (9) and (10) by more than the factors indicated above. We calculate the instantaneous mistake bounds for the Bayes and Gibbs algorithms in the natural case that $\mathcal{F}$ is the set of homogeneous linear threshold functions on $\mathbf{R}^d$ and both the distribution $\mathcal{D}$ and the prior $\mathcal{P}$ on possible target concepts (represented also by vectors in $\mathbf{R}^d$) are uniform on the unit sphere in $\mathbf{R}^d$. This class has VC dimension $d$. In this case, under certain reasonable assumptions used in statistical mechanics, it can be shown that for $m \gg d \gg 1$,

$$\mathbf{E}_{f \in \mathcal{P}, \mathbf{x} \in \mathcal{D}} \cdot [Bayes_m(\mathbf{x}, f)] \approx \frac{0.44d}{m}$$

(compared with the upper bound of $d/m$ given by Inequality (9) for any class of VC dimension $d$) and

$$\mathbf{E}_{f \in \mathcal{P}, \mathbf{x} \in \mathcal{D}} \cdot [Gibbs_m(\mathbf{x}, f)] \approx \frac{0.62d}{m}$$

(compared with the upper bound of $2d/m$ in Inequality (10)). The ratio of these asymptotic bounds is $\sqrt{2}$. We can also show that this performance advantage of Bayes over Gibbs is quite robust even when $\mathcal{P}$ and $\mathcal{D}$ vary, and there is noise in the examples [6].

## 7    OTHER RESULTS AND CONCLUSIONS

We have a number of other results, and briefly describe here one that may be of particular interest to neural network researchers. In the case that the class $\mathcal{F}$ has infinite VC dimension (for instance, if $\mathcal{F}$ is the class of all multi-layer perceptrons of finite size), we can still obtain bounds on the number of cumulative mistakes by decomposing $\mathcal{F}$ into $\mathcal{F}_1, \mathcal{F}_2, \ldots, \mathcal{F}_i, \ldots$, where each $\mathcal{F}_i$ has finite VC dimension, and by decomposing the prior $\mathcal{P}$ over $\mathcal{F}$ as a linear sum $\mathcal{P} = \sum_{i=1}^{\infty} \alpha_i \mathcal{P}_i$, where each $\mathcal{P}_i$ is an arbitrary prior over $\mathcal{F}_i$, and $\sum_{i=1}^{\infty} \alpha_i = 1$. A typical decomposition might let $\mathcal{F}_i$ be all multi-layer perceptrons of a given architecture with at most $i$ weights, in which case $d_i = O(i \log i)$ [1]. Here we can show an upper bound on the cumulative mistakes during the first $m$ examples of roughly $\mathcal{H}\{\alpha_i\} + [\sum_{i=1}^{\infty} \alpha_i d_i] \log m$ for both the Bayes and Gibbs algorithms, where $\mathcal{H}\{\alpha_i\} = -\sum_{i=1}^{\infty} \alpha_i \log \alpha_i$. The quantity $\sum_{i=1}^{\infty} \alpha_i d_i$ plays the role of an "effective VC dimension" relative to the prior weights $\{\alpha_i\}$. In the case that $\mathbf{x}$ is also chosen randomly, we can bound the probability of mistake on the $m$th trial by roughly $\frac{1}{m}(\mathcal{H}\{\alpha_i\} + [\sum_{i=1}^{\infty} \alpha_i d_i] \log m)$.

In our current research we are working on extending the basic theory presented here to the problems of learning with noise (see Opper and Haussler [6]), learning multi-valued functions, and learning with other loss functions.

Perhaps the most important general conclusion to be drawn from the work presented here is that the various theories of learning curves based on diverse ideas from information theory, statistical physics and the VC dimension are all in fact closely related, and can be naturally and beneficially placed in a common Bayesian framework.

## Acknowledgements

We are greatly indebted to Ron Rivest for his valuable suggestions and guidance, and to Sara Solla and Naftali Tishby for insightful ideas in the early stages of this investigation. We also thank Andrew Barron, Andy Kahn, Nick Littlestone, Phil Long, Terry Sejnowski and Haim Sompolinsky for stimulating discussions on these topics. This research was supported by ONR grant N00014-91-J-1162, AFOSR grant AFOSR-89-0506, ARO grant DAAL03-86-K-0171, DARPA contract N00014-89-J-1988, and a grant from the Siemens Corporation. This research was conducted in part while M. Kearns was at the M.I.T. Laboratory for Computer Science and the International Computer Science Institute, and while R. Schapire was at the M.I.T. Laboratory for Computer Science and Harvard University.

## Footnotes

*Contact author. Address: AT&T Bell Laboratories, 600 Mountain Avenue, Room 2A-423, Murray Hill, New Jersey 07974. Electronic mail: mkearns@research.att.com.

[1]It follows that the expected total number of mistakes of the Bayes and the Gibbs algorithms differ by a factor of at most about 1.44 in each of these cases; this was not previously known.

## References

[1] E. Baum and D. Haussler. What size net gives valid generalization? *Neural Computation*, 1(1):151–160, 1989.

[2] J. Denker, D. Schwartz, B. Wittner, S. Solla, R. Howard, L. Jackel, and J. Hopfield. Automatic learning, rule extraction and generalization. *Complex Systems*, 1:877–922, 1987.

[3] G. Györgi and N. Tishby. Statistical theory of learning a rule. In *Neural Networks and Spin Glasses*. World Scientific, 1990.

[4] D. Haussler, M. Kearns, and R. Schapire. Bounds on the sample complexity of Bayesian learning using information theory and the VC dimension. In *Computational Learning Theory: Proceedings of the Fourth Annual Workshop*. Morgan Kaufmann, 1991.

[5] D. Haussler, N. Littlestone, and M. Warmuth. Predicting $\{0, 1\}$-functions on randomly drawn points. Technical Report UCSC-CRL-90-54, University of California Santa Cruz, Computer Research Laboratory, Dec. 1990.

[6] M. Opper and D. Haussler. Calculation of the learning curve of Bayes optimal classification algorithm for learning a perceptron with noise. In *Computational Learning Theory: Proceedings of the Fourth Annual Workshop*. Morgan Kaufmann, 1991.

[7] N. Sauer. On the density of families of sets. *Journal of Combinatorial Theory (Series A)*, 13:145–147, 1972.

[8] H. Sompolinsky, N. Tishby, and H. Seung. Learning from examples in large neural networks. *Physics Review Letters*, 65:1683–1686, 1990.

[9] N. Tishby, E. Levin, and S. Solla. Consistent inference of probabilities in layered networks: predictions and generalizations. In *IJCNN International Joint Conference on Neural Networks*, volume II, pages 403–409. IEEE, 1989.

[10] V. N. Vapnik. *Estimation of Dependences Based on Empirical Data*. Springer-Verlag, New York, 1982.

[11] V. N. Vapnik and A. Y. Chervonenkis. On the uniform convergence of relative frequencies of events to their probabilities. *Theory of Probability and its Applications*, 16(2):264–80, 1971.